# Training Binary Neural Networks
# via Gaussian Variational Inference
# and Low-Rank Semidefinite Programming

**Lorenzo Orecchia**
University of Chicago

**Jiawei Hu**
Georgia Institute of Technology

**Xue He**
Northeastern University of China

**Zhe Wang**
I²R, A∗STAR

**Xulei Yang**
I²R, A∗STAR

**Min Wu**∗
I²R, A∗STAR

**Xue Geng**
I²R, A∗STAR

## Abstract

Improving the training of Binarized Neural Networks (BNNs) is a longstanding challenge whose outcome can significantly affect our ability to deploy deep learning ubiquitously. Current methods heavily rely on latent weights and the heuristic *straight-through estimator* (STE), which enable the application of SGD-based optimizers to the combinatorial training problem, but remain theoretically poorly understood. In this paper, we propose an optimization framework for BNN training based on Gaussian variational inference. Our approach yields a non-convex linear programming formulation that *theoretically motivates the use of latent weights, STE and weight clipping*. More importantly, it allows us to *go beyond latent weights* to formulate and solve low-rank semidefinite programming (SDP) relaxations that explicitly *model and learn pairwise correlations between weights during training*, resulting in improved accuracy. Our empirical evaluation on CIFAR-10, CIFAR-100, Tiny-ImageNet and ImageNet datasets shows our method consistently outperforms all state-of-the-art algorithms for training BNNs.

## 1 Introduction

The advent of deep learning has revolutionized the field of machine learning and enabled stunning technological advances in numerous application areas, including computer vision [He et al., 2017], speech recognition [Baevski et al., 2020] and natural language processing [Devlin et al., 2018]. Despite these achievements, the broader application of deep learning is impeded by high computational demands, requiring the advanced hardware and energy consumption typically reserved for supercomputers [Thompson et al., 2022], both for training and inference. This barrier is particularly formidable when deep learning is deployed on resource-constrained devices, like smartphones and IoT devices, where limitations in memory, processing power and energy are critical [Sze et al., 2017].

To address these issues, Muller and Indiveri [2015] and Courbariaux et al. [2015b] noted that neural networks could provide the same level of performance while restricting the precision of the representation of parameters to a small number of bits, effectively quantizing the space of weights. To reap the full benefits of quantization, Courbariaux et al. [2015b, 2016] and Kim and Smaragdis [2016] independently introduced *binarized neural networks* (BNNs), which use 1-bit representations for each weight, directly leading to a 32-fold reduction in model size compared to single-precision weights. When activation is further binarized [Hubara et al., 2016], the multiplication and addition operations can be replaced by much faster and cheaper XNOR and popcount operations, resulting in

---

∗Corresponding authors

significant cost reduction in both memory and computation. Despite their compelling computational advantages, the problem of constructing and training high-performing BNNs is still open, as current approaches still yield severe accuracy loss compared to their high-precision counterparts [Rastegari et al., 2016, Liu et al., 2018].

**BNN training**  While substantial effort has been aimed at constructing larger and more effective BNN architectures [Umuroglu et al., 2017, Tang et al., 2017, Liu et al., 2020, Martinez et al., 2020, Shen et al., 2020], training BNNs has remained a significant challenge, as the binary constraints yield an intrinsically combinatorial optimization problem, which is a poor fit for traditional continuous optimizers like SGD and Adam. For a domain set $\mathcal{X}$ and a label set $\mathcal{Y}$, let $f : \mathcal{X} \times \mathbb{R}^n \to \mathcal{Y}$ represent a neural network with an $n$-dimensional weight vector. Denote by $y_{\boldsymbol{x}}$ the true label of instance $\boldsymbol{x}$ and by $L$ the smooth loss function. The BNN training problem can then be formulated as the following stochastic optimization problem over the hypercube of binary weights $\hat{\boldsymbol{w}}$:

$$\min_{\hat{\boldsymbol{w}} \in \{\pm 1\}^n} \mathbb{E}_{\boldsymbol{x}}[L(f(\boldsymbol{x}, \hat{\boldsymbol{w}}), y_{\boldsymbol{x}})], \tag{1}$$

where $\mathbb{E}_{\boldsymbol{x}}$ denotes the expectation over $\boldsymbol{x}$ uniformly distributed over $\mathcal{X}$. This computational problem adds to the challenge of non-convexity, the additional obstacle of a combinatorial feasible set, making continuous gradient queries potentially uninformative. Moreover, as the training must be carried out over a BNN architicture, we face the *additional restriction that the gradient* $\nabla_{\hat{\boldsymbol{w}}} L(f(\boldsymbol{x}, \hat{\boldsymbol{w}}), y_{\boldsymbol{x}})$ *of the loss can only be evaluated at binary weights* $\hat{\boldsymbol{w}} \in \{\pm 1\}^n$. This further limits our capability to explore the loss landscape.

**Latent Weights**  Currently, most BNN-training procedures make use of latent real weights $\boldsymbol{w} \in \mathbb{R}^n$ [Courbariaux et al., 2015a, 2016] to maintain the state of an iterative training algorithm and guide the optimization process. Latent weights are rounded to binary weights $\hat{\boldsymbol{w}} = \textbf{round}(\boldsymbol{w})$ via a potentially stochastic function $\textbf{round} : \mathbb{R}^n \to \{\pm 1\}^n$ to compute forward and backward passes over the BNN architecture. The most common choice of $\textbf{round}$ function is simply the deterministic $\textbf{sign}$ function applied to each latent weight. Unfortunately, any non-trivial $\textbf{round}$ function is discontinuous and cannot be differentiable over $\mathbb{R}^n$, so that it is not possible to evaluate the true gradient of the loss function with respect to the latent weights $\boldsymbol{w}$. The main practical solution to this problem has been to *simply ignore the* $\textbf{round}$ *function in the back-propagation of the gradient*, yielding the *straight-through estimator* (STE) [Bengio et al., 2013, Le et al., 2022]:

$$\nabla_{\boldsymbol{w}} L(f(\boldsymbol{x}, \textbf{round}(\boldsymbol{w})), y_{\boldsymbol{x}}) \approx \nabla_{\hat{\boldsymbol{w}}} L(f(\boldsymbol{x}, \hat{\boldsymbol{w}}), y_{\boldsymbol{x}})|_{\hat{\boldsymbol{w}} = \textbf{round}(\boldsymbol{w})} \tag{STE}$$

While there is no theoretical assurance that the STE is a valid proxy for the gradient [Yin et al., 2019], the STE and its variants [Le et al., 2022, Wu et al., 2023] have proved remarkably effective in practice, particularly in combination with *weight clipping* [Alizadeh et al., 2018, Merolla et al., 2016], by which latent weights are constrained to a fixed range around the origin.

**Theoretical Interpretations of Latent-Weights Methods**  Latent weights have usually been interpreted as fractional approximations of the true binary weights [Anderson and Berg, 2017]. The influential work of Helwegen et al. [2019] instead proposes to view latent weights as a measurement of the algorithm's confidence in the binary weight taking on a certain sign. With this intuition, Ajanthan et al. [2019] and Meng et al. [2020] have suggested a more formal interpretation of latent weights based on the mean-field approximation from variational inference [Wainwright et al., 2008]. In this setup, which we review in Section 3, the latent weights are the mean parameters of an exponential family of probability distributions over binary weights. However, none of these works are able to fully justify the use of the STE and weight clipping in latent-weights methods.

**Our contributions**  In this paper, we provide a general optimization framework for BNN training based on Gaussian variational inference, a refinement of the mean-field approximation used by Ajanthan et al. [2019] and Meng et al. [2020]. *Our framework allows us to generalize the notion of latent weights as mean parameters in order to introduce new variables modeling the covariances of the weights.* As a result, our optimization formulation is able to capture more intricate dependencies among weights and exploit them to learn better solutions during the training phase. We believe the mere statement of our formulation in Section 3.1 to be a significant contribution of our work, which will hopefully lead to further study of Gaussian variational methods for neural network training.

In Section 3.2, we show that the general case of our framework, i.e., the problem of learning an optimal Gaussian distribution over weights in $\mathbb{R}^n$, can be cast as a non-convex semi-definite program (SDP) over the weights mean vector $\boldsymbol{\mu} \in \mathbb{R}^n$ and covariance matrix $\boldsymbol{\Sigma} \in \mathbb{R}^{n \times n}$. Because of the large number of weights $n$ in typical applications, we do not attempt to maintain full-rank covariance matrices, but only consider low-rank approximate SDP solutions, following the approach championed by Burer and Monteiro [2005] for linear SDP programs. *The resulting algorithm (Algorithm 1) is the main contribution of this paper.* We name our method the Variational Inference Semidefinite Programming Algorithm (VISPA).

In Section 3.3, we demonstrate that a simpler instantiation of our framework yields a non-convex program, whose solution by gradient descent *naturally recovers the use of the STE and weight clipping* in latent-weights methods. Finally, in Section 4, we present a thorough experimental evaluation of VISPA against a large number of BNN training procedures in the literature over four standard benchmark datasets. We find that VISPA *almost always improves the state-of-the-art accuracy of BNN training*, in some cases dramatically. For instance, Top-1 accuracy with AlexNet on ImageNet with fully binarized weights and activations increases by more than $3\%$ compared to the state-of-the-art method (see Table 3). Through an ablation study, we also show that the SDP component of our algorithm is crucial in realizing the observed empirical advantage. Code for our algorithm and experimental evaluation can be found at `https://github.com/snownus/bnn_vi`.

**Limitations and Open Problems**  We focus our first presentation of VISPA on vision-based applications because of the availability of well-studied binarized architectures and well-established baselines, which isolate the performance of our method more closely. Indeed, in the case of transformers, there is not yet agreement on the best binarized architecture, due to the difficulty of binarizing activations in softmax layers. Only recently, researchers have made progress in bypassing this obstacle He et al. [2023]. This also contributes to the scarcity of baselines and the absence of a standard benchmark. In Section 5, we discuss other limitations of the current work and opportunities for future extensions.

## 2   Related Work

Courbariaux et al. [2015a] introduced the use of the STE for training BNNs. Since then, researchers have put forward many variants to this idea, such as adaptive versions of the STE [Le et al., 2022, Wu et al., 2023, Qin et al., 2023], and a number of extensions, e.g., to non-binary quantization [Huh et al., 2023, Liu et al., 2024] and sparsity-driven network designs [Vanderschueren and De Vleeschouwer, 2023]. All of these variations are in principle applicable within our optimization framework.

A different line of work investigates alternatives to the STE, with two approaches standing out. Modifications to the gradient estimator include using piecewise polynomial functions (BiRealnet [Liu et al., 2018]) and dynamic gradient estimators (IR-Net [Qin et al., 2020], RBNN [Lin et al., 2020]). The other approach designs separate frameworks for discrete back-propagation (PCNN [Gu et al., 2019a], BiPer [Vargas et al., 2024], ReCU [Xu et al., 2021b], ReActNet [Liu et al., 2020]). Among this latter class, the aforementioned work of Helwegen et al. [2019], followed by several variants [Suarez-Ramirez et al., 2021, Shan et al., 2023], proposes a novel Binary Optimizer (Bop) that maintains a binary solution and accumulates gradients to determine when to flip a bit. The only BNN-training method based on variational inference, BayesBiNN by Meng et al. [2020], effectively combines the STE and Bop in a principled way. The work of Ajanthan et al. [2019], which is also based on variational inference, only deals with network quantization and does not perform training over a BNN architecture. Of particular relevance to this paper is the LNS algorithm of Han et al. [2020], who also notice that simply binarizing each latent weight independently does not fully explore the relationship between neurons and may not lead to the optimal solution. They propose to train a custom binarization function via supervision noise learning, but do not explicitly model correlations between weights via new variables. Our experimental evaluation compares our algorithm with all the methods just described, showing the superior accuracy of our technique in practice.

Since the seminal work of Goemans and Williamson [1995], semidefinite programming [Vandenberghe and Boyd, 1996] has become a fundamental tool in the design of approximation algorithms for combinatorial optimization problems. Recently, its application to network quantization has been studied by Bartan and Pilanci [2021]. They construct a tight SDP relaxation for training a two-layer quantized neural network. Crucially, their method does not run the training on the quantized architecture, i.e., network gradients are evaluated at non-quantized weight settings, and are limited to

small-scale, shallow networks. However, we believe their idea provides valuable theoretical evidence in favor of the deployment of SDP techniques at a larger scale, as is done in our work.

# 3 Variational Inference Approach to BNN Training

We start by quickly reviewing the variational inference approach to BNN training before introducing our novel contribution in Section 3.1. A common approach to the construction of approximation algorithms for intractable combinatorial optimization tasks [Vazirani, 2010, Barak and Steurer, 2024] is to consider relaxed, regularized formulations over a subset $\mathcal{P}$ of the simplex $\Delta_n = \{\boldsymbol{p} : \{\pm 1\}^n \to \mathbb{R}_{\geq 0}, \sum_{\hat{\boldsymbol{w}} \in \{\pm 1\}^n} \boldsymbol{p}_{\hat{\boldsymbol{w}}} = 1\}$ of probability distributions over the hypercube $\{\pm 1\}^n$:

$$\text{OPT}_{\mathcal{P}} = \min_{\boldsymbol{p} \in \mathcal{P}} \mathbb{E}_{\hat{\boldsymbol{w}} \sim \boldsymbol{p}, \boldsymbol{x}}[L(f(\boldsymbol{x}, \hat{\boldsymbol{w}}), y_{\boldsymbol{x}})] - \lambda \cdot H(\boldsymbol{p}), \tag{2}$$

where $\hat{\boldsymbol{w}} \sim \boldsymbol{p}$ indicates that the random variable $\hat{\boldsymbol{w}}$ is distributed according to the distribution $\boldsymbol{p}$ and $H$ denotes the entropy function. When the regularization parameter $\lambda \geq 0$ is strictly positive, the regularization term $-\lambda \cdot H$ is known to encourage generalization. When $\mathcal{P}$ equals the set of all probability distributions over $\{\pm 1\}^n$, this relaxation renders the loss term linear in the distribution $\boldsymbol{p}$, but requires an exponential-size representation, hence maintaining the computational hardness of the problem. However, this formulation allows us to reason more directly about stochastic approaches, such as Monte Carlo Markov Chain [Gamerman and Lopes, 2006] and variational inference methods [Wainwright et al., 2008]. In particular, the latter approach suggests restricting the feasible space of Problem 2 to a computationally tractable class of distributions $\mathcal{P}$, such as an exponential family, in order to obtain a more compact parametrization of a space of probabilities. We can then attempt to find an approximately optimal solution to the resulting non-convex optimization problem via gradient descent over the distribution parameters.

Previous works by Ajanthan et al. [2019] and Meng et al. [2020] provide a more rigorous justification for the STE step by deploying this variational inference blueprint in the form of the well-known mean-field approximation [Friedli and Velenik, 2017, Sayama, 2015]. Specifically, they restrict $\mathcal{P}$ to a product of $\{\pm 1\}$-Bernoulli distributions, one for each coordinate, where $\boldsymbol{p}_i$ equals the probability that $\hat{\boldsymbol{w}}_i$ equals 1. The distribution of the variable $\hat{\boldsymbol{w}}_i$ can then be re-parametrized in terms of its mean $\boldsymbol{\mu}_i$ as, for all $i$,

$$\hat{\boldsymbol{w}}_i \sim \text{Bernoulli}(\boldsymbol{p}_i), \quad \boldsymbol{p}_i = \frac{1 + \boldsymbol{\mu}_i}{2}, \quad \boldsymbol{\mu}_i \in [-1, 1]. \tag{$\mathcal{P}_{\text{Bernoulli}}$}$$

Notice that Problem 2 with $\mathcal{P} = \mathcal{P}_{\text{Bernoulli}}$ is still a relaxation to the original Problem 1, as the extreme values of $\boldsymbol{\mu}$ yield deterministic weight choices. At this point, Ajanthan et al. [2019] and Meng et al. [2020] then argue that the vector of mean parameter $\boldsymbol{\mu}$ constitutes the right choice of latent weights for the BNN training problem. In this way, the fixed range $[-1, +1]^n$ of $\boldsymbol{\mu}$ explains the use of weight clipping. However, the main challenge with this approach is the estimation of the gradient $\nabla_{\boldsymbol{\mu}} \mathbb{E}_{\boldsymbol{w} \sim p(\boldsymbol{\mu}), \boldsymbol{x}}[L(f(\boldsymbol{x}, \boldsymbol{w}), y_{\boldsymbol{x}})]$ of the expected loss with respect to the mean parameters $\boldsymbol{\mu}$. In their case, one can only establish the following general form [Williams, 1992]:

$$\nabla_{\boldsymbol{\mu}} \mathbb{E}_{\boldsymbol{w} \sim p(\boldsymbol{\mu}), \boldsymbol{x}}[L(f(\boldsymbol{x}, \boldsymbol{w}), y_{\boldsymbol{x}})] = \mathbb{E}_{\boldsymbol{w} \sim p(\boldsymbol{\mu}), \boldsymbol{x}}[L(f(\boldsymbol{x}, \boldsymbol{w}), y_{\boldsymbol{x}}) \nabla_{\boldsymbol{\mu}} \log \boldsymbol{p}(\boldsymbol{w})].$$

This expression leads to estimating the gradient via sampling from $\boldsymbol{p}$ and evaluating the loss function, but fails to take advantage of the differentiability of $\mathcal{L}$ and fails to reproduce the STE. Indeed, this setback forces Meng et al. [2020] to use a smooth proxy to the **sign** function to recover an approximation of the STE. The work of Ajanthan et al. [2019] only performs binary compression and does not rely on the STE.

## 3.1 BNN Training via Gaussian Variational Inference

In this section, we describe our novel application of Gaussian variational inference, which has long been recognized as the most practical refinement of the mean-field approach [Giordano et al., 2015], to BNN training. Specifically, we consider optimizing Problem 2 over the class $\mathcal{P}_{\text{corr}}$ of correlated multivariate Gaussian distributions over $\mathbb{R}^n$, including degenerate Gaussian distributions with rank-deficient covariance matrices:

$$\boldsymbol{w} \sim \mathcal{N}(\boldsymbol{\mu}, \boldsymbol{\Sigma}), \quad \boldsymbol{\mu} \in [-1, 1]^n, \quad \boldsymbol{\Sigma} \succeq 0, \tag{$\mathcal{P}_{\text{corr}}$}$$
$$\forall i \in [n], \boldsymbol{\Sigma}_{ii} + \boldsymbol{\mu}_i^2 = 1.$$

---

**Algorithm 1** BNN Training Algorithm `VISPA`

---

**Input**: Loss Function $L(f(\boldsymbol{x}, \boldsymbol{w}), y_{\boldsymbol{x}})$, Batch Size $M$
**Parameter**: Embedding Dimension $K$, Weight Mean Vector Initialization $\boldsymbol{\mu} \in \mathbb{R}^n$, Weight Deviation Matrix $\boldsymbol{Z} \in \mathbb{R}^{n \times K}$, Step Length $\alpha$, Momentum Coefficient $\beta$, Number of Epochs $T$
**Output**: Learned Weight Mean Vectors $\boldsymbol{\mu} \in \mathbb{R}^n$ and Weight Deviation Matrix $\boldsymbol{Z} \in \mathbb{R}^{n \times K}$

---

1: **while** number of epochs less than $T$ **do**
2:     Sample $\boldsymbol{r} \in \mathbb{R}^K$ from $N(0, \boldsymbol{I})$;                        $\triangleright$ Sample rounding vector
3:     Sample mini-batch $(\boldsymbol{x}_1, y_{\boldsymbol{x}_1}), \ldots, (\boldsymbol{x}_M, y_{\boldsymbol{x}_M})$;
4:     $\boldsymbol{w} = \boldsymbol{\mu} + \boldsymbol{Z}\boldsymbol{r}$;                           $\triangleright$ Sample Gaussian weights
5:     $\hat{\boldsymbol{w}} = \mathbf{sign}(\boldsymbol{w})$;                          $\triangleright$ Round to binary
6:

$$\boldsymbol{g} = \frac{1}{M} \sum_{m=1}^{M} \nabla_{\boldsymbol{w}} L(f(\boldsymbol{x}_m, \boldsymbol{w}), y_{\boldsymbol{x}_m})|_{\boldsymbol{w} = \hat{\boldsymbol{w}}}; \qquad (\triangleright \text{Estimate gradient of vector embedding})$$

7:     $\boldsymbol{\mu}_v = \beta\boldsymbol{\mu}_v + (1-\beta)\boldsymbol{g}$;                $\triangleright$ Update velocity for $\boldsymbol{\mu}$
8:     $\boldsymbol{Z}_v = \beta\boldsymbol{Z}_v + (1-\beta)(\boldsymbol{g}\boldsymbol{r}^T)$;           $\triangleright$ Update velocity for $\boldsymbol{Z}$
9:     $\boldsymbol{\mu} = \boldsymbol{\mu} - \alpha\boldsymbol{\mu}_v$;            $\triangleright$ Update weight mean vector with momentum
10:     $\boldsymbol{Z} = \boldsymbol{Z} - \alpha\boldsymbol{Z}_v$;         $\triangleright$ Update weight deviation vector with momentum
11:     For all $i \in [n], \boldsymbol{\gamma}_i = \boldsymbol{\mu}_i^2 + (\boldsymbol{Z}\boldsymbol{Z}^T)_{ii}$;        $\triangleright$ Compute normalization factor
12:     For all $i \in [n], \boldsymbol{\mu}_i = \frac{1}{\sqrt{\gamma_i}}\boldsymbol{\mu}_i$;         $\triangleright$ Normalize weight mean vector
13:     For all $i \in [n], \boldsymbol{z}_i = \frac{1}{\sqrt{\gamma_i}}\boldsymbol{z}_i$;         $\triangleright$ Normalize weight variance vector
14: **end while**
15: **return** $\boldsymbol{\mu}, \boldsymbol{Z}$

---

The joint constraints on mean and covariance ensure that the second moments $\mathbb{E}[\boldsymbol{w}_i^2]$ equal 1, matching those of a distribution over $\{\pm 1\}^n$. The resulting non-convex semidefinite program is also a valid relaxation of Problem 1, as setting $\boldsymbol{\Sigma} = 0$ yields $\boldsymbol{\mu} \in \{\pm 1\}^n$.

At first, the relaxation of the sample space of $\mathcal{P}_{\text{corr}}$ from $\{\pm 1\}$ to $\mathbb{R}^n$ may seem problematic, as sampling now fails to yield the desired binary weights. However, the use of multivariate Gaussians as tractable proxies for the discrete probability distributions in $\mathcal{P}_{\text{Bernoulli}}$ has a long history in approximation algorithms, particularly in the context of semidefinite programming relaxations [Goemans and Williamson, 1995, Alon and Naor, 2006]. Indeed, the celebrated Grothendieck's inequality [Grothendieck, 1953] shows that $\mathcal{P}_{\text{Bernoulli}}$ can be effectively relaxed to $\mathcal{P}_{\text{corr}}$, with only a multiplicative constant loss, when optimizing the expectation of a quadratic polynomial. Unfortunately, the corresponding rounding procedure from a sample $\boldsymbol{w} \sim \boldsymbol{p}, \boldsymbol{p} \in \mathcal{P}_{\text{corr}}$ to a binary vector $\hat{\boldsymbol{w}}$ is fairly complex, as it requires taking large tensor powers of the entries of the covariance of $\boldsymbol{p}$ [Alon and Naor, 2006]. We opt instead for the more straightforward *hyperplane rounding* [Goemans and Williamson, 1995], which takes the simple form $\hat{\boldsymbol{w}} = \mathbf{sign}(\boldsymbol{w})$, recovering the standard sign-based rounding. In this case, the approximation guarantee only holds for quadratic polynomials with non-negative coefficients. This still provides sufficient theoretical motivation for our method and enables the higher performance of our algorithms, as practical results in Section 4 demonstrate.

### 3.2 Solving the SDP over Low-Rank Covariances

In this subsection, we describe `VISPA`, our algorithm for solving the SDP formulation of Problem 2 over $\mathcal{P} = \mathcal{P}_{\text{corr}}$. Inspired by the previous discussion on hyperplane rounding, we let $\lambda$ go to 0, so that no entropy regularization is performed, but the SDP formulation captures more closely the original Problem 2. It is crucial to notice that we cannot hope to maintain a general covariance matrix $\boldsymbol{\Sigma} \in \mathbb{R}^{n \times n}, \boldsymbol{\Sigma} \succeq 0$, as this requires storing $\Omega(n^2)$ matrix entries, which is infeasible for the large number of weights ($n >> 10^6$) in practical BNN architectures. Fortunately, this is a typical issue with large-scale SDPs [Yurtsever et al., 2021], which can be tackled by restricting our attention to low-rank covariance solutions, as first suggested by Burer and Monteiro [2005]. Following this setup, for a rank parameter $K \in \mathbb{N}$, our algorithm maintains a vector of means $\boldsymbol{\mu} \in \mathbb{R}^n$ and a square root $\boldsymbol{Z} \in \mathbb{R}^{n \times K}$ of the covariance $\boldsymbol{\Sigma} = \boldsymbol{Z}\boldsymbol{Z}^T$, which is now of rank at most $K$. We call $\boldsymbol{Z}$ a *weight*

*deviation matrix*, as it describes the typical deviation from the mean $\boldsymbol{\mu}$ along the subspace identified by the image of $\boldsymbol{\Sigma}$. Pseudocode for the resulting algorithm `VISPA` is given in Algorithm 1. Here, we immediately see an advantage of the parametrization by the square root $\boldsymbol{Z}$ of $\boldsymbol{\Sigma}$ rather than by $\boldsymbol{\Sigma}$ itself: it facilitates sampling from the underlying Gaussian distribution, as a sample $\boldsymbol{w}$ can be easily taken by rescaling a standard $K$-dimensional Gaussian $\boldsymbol{r}$ (line 4 of Algorithm 1).

In contrast with classical applications of semidefinite programming, the objective function for the SDP of interest is nonlinear and non-convex, so that we must rely on gradient descent to solve the program to local optimality. The following theorem, proved in the Appendix, shows that the gradients of the expected loss with respect to the parameters $\boldsymbol{\mu}$ and $\boldsymbol{Z}$ take on a particularly simple form, which is easy to estimate stochastically. This result, which heavily relies on the Gaussianity of the weights $\boldsymbol{w}$, implicitly solves the challenge encountered by Meng et al. [2020] in previous work.

**Theorem 1.** *For a random variable* $\boldsymbol{w} \sim \mathcal{N}(\boldsymbol{\mu}, \boldsymbol{Z}\boldsymbol{Z}^T)$, *with* $\boldsymbol{\mu} \in \mathbb{R}^n$ *and* $\boldsymbol{Z} \in \mathbb{R}^{n \times K}$, *we have:*

$$\nabla_{\boldsymbol{\mu}} \mathbb{E}_{\boldsymbol{w},\boldsymbol{x}}[L(f(\boldsymbol{x},\boldsymbol{w}), y_{\boldsymbol{x}})] = \mathbb{E}_{\boldsymbol{w},\boldsymbol{x}}[\nabla_{\boldsymbol{w}} L(f(\boldsymbol{x},\boldsymbol{w}), y_{\boldsymbol{x}})];$$

$$\nabla_{\boldsymbol{Z}} \mathbb{E}_{\boldsymbol{w},\boldsymbol{x}}[L(f(\boldsymbol{x},\boldsymbol{w}), y_{\boldsymbol{x}})] = \mathbb{E}_{\boldsymbol{r},\boldsymbol{x}}[\nabla_{\boldsymbol{w}}[L(f(\boldsymbol{x},\boldsymbol{w}), y_{\boldsymbol{x}})\boldsymbol{r}^T]|_{\boldsymbol{w}=\boldsymbol{Z}\boldsymbol{r}+\boldsymbol{\mu}}],$$

*where* $\boldsymbol{r} \sim \mathcal{N}(0, \boldsymbol{I}_K)$.

The approximation provided by hyperplane rounding justifies replacing $\boldsymbol{w}$ with $\mathbf{sign}(\boldsymbol{w})$ in the right-hand side of the gradient expressions in the theorem. The mini-batch stochastic gradient descent step with momentum then takes the form of lines 6-10 in Algorithm 1, with lines 7-8 regulating the momentum. The moment-matching constraints are enforced by the projection steps of lines 11-13. Finally, at the inference stage, the output mean vector $\boldsymbol{\mu}$ and deviation matrix $\boldsymbol{Z}$ are used to sample binary weights via $\mathbf{sign}$ rounding. In Section 4, we carry out a comprehensive evaluation of the accuracy of Algorithm 1 against state-of-the-art methods for BNN training.

**Running Time of `VISPA`**   As in Algorithm 1, we let $n$ be the number of weight parameters and $K$ the embedding dimension. Additionally, denote by $M$ the batch size. The running time of one iteration of `VISPA` is $O(Mn + nK)$, where the first term stems from propagating each example through the neural network and the second term comes from updating the weight mean vector $\mu$ and the weight deviation matrix $\boldsymbol{Z}$.. In comparison, other state-of-the-art BNN training approaches typically just require $O(Mn)$ time. In most cases, we have that $K << M$, so that the increased cost due to the $nK$ term is a negligible fraction of the total running time, as most time is spent performing forward- and back-propagation through the neural network.

**Memory consumption of `VISPA`**   The main potential limitation of VISPA on resource-constrained devices is the increase in memory usage due to having to maintain the covariance variable $\boldsymbol{Z}$. Indeed, `VISPA` requires $n \cdot (K + 1)$ memory for storing relevant variables, compared to just $n$ for other methods. The total significance of this increase depends on the batch size and the size of the examples. For instance, in the case of ResNet18, there are roughly $n = 9 \cdot 10^6$ weights while, for batch size 100, the total size of a data batch is $15 \cdot 10^6$. Hence, standard methods yield a total memory usage of $24 \cdot 10^6$. In contrast, choosing $K = 1$ in `VISPA` will lead to a usage of $33 \cdot 10^6$, a $37.5\%$ increase. This increase will typically get smaller as we consider models trained on larger images. In practice, we often observe even smaller increases, as our estimate does not include memory usage due to back-propagation, which can be very large, e.g., in the case of residual connections. A possible mitigation strategy is to store our variable at a lower precision. Given that this variable is only accessed via multiplication with Gaussian noise, we believe that this will not change the behavior of our algorithm.

### 3.3   Diagonal Covariances and New Interpretation of Latent-Weights Methods

Now, we consider the simple case in which the covariance matrix $\boldsymbol{\Sigma}$ is a multiple $\sigma^2 \boldsymbol{I}$ of the identity, so that the underlying weights $\boldsymbol{w}$ are independent Gaussian random variables with mean $\boldsymbol{\mu}$. We denote the associated class of distributions by $\mathcal{P}_{\text{indep}}$:

$$\boldsymbol{w} \sim \mathcal{N}(\boldsymbol{\mu}, \sigma^2 \boldsymbol{I}), \quad \boldsymbol{\mu} \in [-1, 1]^n \qquad\qquad (\mathcal{P}_{\text{indep}})$$

By applying Theorem 1 in conjunction with hyperplane rounding, the mini-batch stochastic gradient descent step for Problem 2 with $\mathcal{P} = \mathcal{P}_{\text{indep}}$ takes the following form:

$$\forall t \in \mathbb{N}, \quad \boldsymbol{g}^{(t)} = \frac{1}{M} \sum_{m=1}^{M} \nabla L(f(\boldsymbol{x}_m, \textbf{sign}(\boldsymbol{w}^{(t)})), y_{\boldsymbol{x}_m}), \quad \boldsymbol{\mu}^{(t+1)} = \text{clip}(\boldsymbol{\mu}^{(t)} - \alpha \boldsymbol{g}^{(t)}), \quad (3)$$

for a choice of step length $\alpha > 0$, sample mini-batch $(\boldsymbol{x}_1, y_{\boldsymbol{x}_1}), \ldots, (\boldsymbol{x}_M, y_{\boldsymbol{x}_M})$ and a sample $\boldsymbol{w}^{(t)}$ from $\mathcal{P}_{\text{indep}}$ with mean parameter $\boldsymbol{\mu}^{(t)}$. The clip function restricts the argument to the range $[-1, +1]$. Surprisingly, this derivation recovers both the use of the STE and weight clipping, as well as that of the **sign** function for rounding. In Section 4.1, we compare the deterministic version of this algorithm ($\sigma = 0$) with Algorithm 1 to show the accuracy gain due to the use of the deviation matrix $\boldsymbol{Z}$.

## 4    Experiments

**Datasets** We evaluate the performance of various methods on four datasets: CIFAR-10 [Krizhevsky et al., 2009], CIFAR-100 [Krizhevsky et al., 2009], Tiny-ImageNet [Le and Yang, 2015] and ImageNet [Deng et al., 2009]. CIFAR-10 consists of 50k training samples and 10k testing images with 10 classes, while CIFAR-100 consists of 50k training samples and 10k testing images with 100 non-overlapping classes. Tiny-ImageNet is a subset of ImageNet with 100k images and 200 classes. ImageNet contains 1.28 million training samples and 50k testing images for 1000 classes.

**Implementation Details** We implement VISPA using PyTorch [Paszke et al., 2019] and run on a single NVIDIA A100 with 40GB GPU memory per GPU card. On CIFAR-10, CIFAR-100 and Tiny-ImageNet, models with only binarized weights (denoted as 1W32A) are trained for 500 epochs following [Le et al., 2022], using a batch size of 256, an initial learning rate of 0.1, and a weight decay of $5e - 4$, with covariance rank $K$ set to 8. For models with both binarized weights and activations (denoted as 1W1A), the training epochs is 600 following [Xu et al., 2021b] , the initial learning rate is set to 0.5, and the weight decay is reduced to $1e - 5$, with $K$ set to 4. We run 5 runs to report the mean and standard deviation. All experiments are conducted on a single GPU card.

On ImageNet, we train AlexNet [Krizhevsky et al., 2012] for 100 epochs, and ResNet18 [He et al., 2016] for 200 epochs following [Xu et al., 2021b], with a batch size of 1024, and standard preprocessing with random flips and resize in [He et al., 2016]. Models with 1W32A are trained using an initial learning rate of 0.1, and a weight decay of $5e - 5$, with $K$ set at 4. For models with 1W1A, the initial learning rate is set at 0.5, and the weight decay is reduced to $1e - 5$, with $K$ set at 2. All experiments are conducted on four GPU cards.

All runs utilize a cosine annealing learning rate schedule with a 5-epoch warm-up to optimize training. To accelerate convergence, we employ the momentum technique [Sutskever et al., 2013], setting the momentum coefficient to $\beta = 0.9$. To capture the correlation among weights, at the inference stage, we perform 40 samples and average the results to obtain the final prediction. The mean weights $\boldsymbol{\mu}$ and the deviation matrix $\boldsymbol{Z}$ are initialized using a Xavier normal distribution [Glorot and Bengio, 2010] with a mean of 0 and a standard deviation as $s * \sqrt{\frac{2}{\text{fan\_in} + \text{fan\_out}}}$, where fan_in is the number of input units, fan_out is the number of output units and $s$ is the scaling factor. We empirically set $s = 10$ for $\boldsymbol{Z}$ and $s = 1$ for $\boldsymbol{u}$. For a detailed study of the impact of the initialization of $\boldsymbol{Z}$, please refer to the Appendix A.2.

**Comparison on CIFAR-10, CIFAR-100 and Tiny-ImageNet**    We evaluate the performance of the 1W1A and 1W32A settings on CIFAR-10, CIFAR-100 and Tiny-ImageNet. For the 1W1A setting, we compare various methods, including ReSTE [Wu et al., 2023] and DIR-Net [Qin et al., 2023], using the commonly employed VGG-Small [Zhang et al., 2018] and ResNet18 [He et al., 2016] architectures in [Qin et al., 2020, Wu et al., 2023, Xu et al., 2021b, Lin et al., 2022]. Table 1 shows the result. Our method achieves the highest accuracy with 92.7% on VGG-Small and matches the top performance of 92.8% on ResNet18. While our approach only shows marginal improvements over recent methods like ReSTE, the results demonstrates its efficiency in optimizing binarized neural networks. For the 1W32A setting, we compare various approaches, including AdaSTE [Le et al., 2022] and BayesBiNN [Meng et al., 2020], using the commonly employed VGG16 [Simonyan and Zisserman, 2014] and ResNet18 [He et al., 2016] architectures in [Le et al., 2022, Ajanthan et al., 2019]. Table 2 presents the result. Our proposed method achieves the highest accuracy across all

Table 1: Performance comparison by testing accuracy (%) on CIFAR-10 using VGG-Small and ResNet18 architectures with binarized activations and weights.

| Methods | VGG-Small | ResNet18 |
|---|---|---|
| IR-Net Qin et al. [2020] | 90.4 | 91.5 |
| SD-BNN Xue et al. [2022] | 90.8 | 92.5 |
| RBNN Lin et al. [2020] | 91.3 | 92.2 |
| ReCU Xu et al. [2021b] | 92.2 | **92.8** |
| LCR-BNN Shang et al. [2022a] | – | 91.8 |
| FDA-BNN Xu et al. [2021a] | 92.5 | – |
| RBNN + CMIM Shang et al. [2022b] | 92.2 | **92.8** |
| SiMaN Lin et al. [2022] | 92.5 | 92.5 |
| ReSTE Wu et al. [2023] | 92.6 | 92.6 |
| DIR-Net Qin et al. [2023] | $91.1 \pm 0.1$ | $92.8 \pm 0.1$ |
| VISPA (Ours) | $\mathbf{92.7 \pm 0.1}$ | $\mathbf{92.8 \pm 0.2}$ |

Table 2: Performance comparison by testing accuracy (%) of various approaches on CIFAR-10, CIFAR-100, and Tiny-ImageNet across VGG16 and ResNet18 architectures with binarized weights only. (†) indicates that results are obtained from the numbers reported by Ajanthan et al. [2021]. (*) indicates that results are obtained from the numbers reported by Le et al. [2022].

| Methods | CIFAR-10 | | CIFAR-100 | | Tiny-ImageNet |
|---|---|---|---|---|---|
| | VGG16 | ResNet18 | VGG16 | ResNet18 | ResNet18 |
| BinaryConnect (†) Courbariaux et al. [2015a] | 89.04 | 91.64 | 59.13 | 72.14 | 49.65 |
| ProxQuant (†) Bai et al. [2018] | 90.11 | 92.32 | 55.10 | 68.35 | 49.97 |
| MDS-softmax-s (†) Ajanthan et al. [2021] | 91.30 | 93.28 | 63.97 | 72.18 | 51.81 |
| MDS-tanh-s (†) Ajanthan et al. [2021] | 91.53 | 93.18 | 61.69 | 72.18 | 52.32 |
| PMF (†) Ajanthan et al. [2019] | 91.40 | 93.24 | 64.71 | 71.56 | 51.52 |
| BayesBiNN (*) Meng et al. [2020] | $90.68 \pm 0.07$ | $92.28 \pm 0.09$ | $65.92 \pm 0.18$ | $70.33 \pm 0.25$ | 54.22 |
| AdaSTE (*) Le et al. [2022] | $92.37 \pm 0.09$ | $94.11 \pm 0.08$ | $69.28 \pm 0.17$ | $75.03 \pm 0.35$ | 54.92 |
| VISPA (Ours) | $\mathbf{93.25 \pm 0.11}$ | $\mathbf{95.05 \pm 0.10}$ | $\mathbf{72.09 \pm 0.17}$ | $\mathbf{77.05 \pm 0.41}$ | $\mathbf{58.98 \pm 0.28}$ |

datasets and architectures, outperforming existing state-of-the-art techniques. Specifically, on more complex datasets, our method shows significant improvements. For CIFAR-100, our method reaches 72.09% on VGG16, outperforming the best baseline, AdaSTE, by 2.81%. For Tiny-ImageNet, our approach achieves 58.98% on ResNet18, which is 4.06% higher than AdaSTE. These results highlight the effectiveness of our method in handling more complex datasets.

Table 3: Performance comparison by testing accuracy of various methods on ImageNet dataset at AlexNet. W/A denotes the bit-width of weights and activations.

| Methods | W/A | AlexNet | |
|---|---|---|---|
| | | Top1 (%) | Top5 (%) |
| BinaryNet Hubara et al. [2016] | 1/1 | 41.2 | 65.6 |
| XNOR-Net Rastegari et al. [2016] | 1/1 | 44.2 | 69.2 |
| Bop Helwegen et al. [2019] | 1/1 | 45.9 | 70.0 |
| Bop2ndOrder Suarez-Ramirez et al. [2021] | 1/1 | 46.9 | 70.9 |
| LNS Han et al. [2020] | 1/1 | 44.4 | - |
| FDA-BNN Xu et al. [2021a] | 1/1 | 46.2 | 69.7 |
| Quantization networks Yang et al. [2019] | 1/1 | 47.9 | 72.5 |
| BNN-DL Ding et al. [2019] | 1/1 | 47.8 | 71.5 |
| VISPA (Ours) | 1/1 | **51.1** | **75.0** |
| BinaryConnect Courbariaux et al. [2015a] | 1/32 | 35.4 | 61.0 |
| DoReFa Zhu et al. [2016] | 1/32 | 53.9 | 76.3 |
| XNOR-Net Rastegari et al. [2016] | 1/32 | 56.8 | 79.4 |
| ADMM Leng et al. [2018] | 1/32 | 57.0 | 79.7 |
| Quantization networks Yang et al. [2019] | 1/32 | 58.8 | **81.7** |
| VISPA (Ours) | 1/32 | **59.4** | 81.1 |

**Comparison on ImageNet**  To evaluate the performance of our proposed binarized neural network method, we compare a list of SOTA methods including ReBNN [Xu et al., 2023], ReSTE [Wu et al., 2023], DIR-Net [Qin et al., 2023] and BiPer [Vargas et al., 2024] on the ImageNet dataset using AlexNet and ResNet18 architectures. For AlexNet, we used the standard architecture without binarizing the first and last layers, adapting it for 1W1A and 1W32A configurations. For ResNet18, we employed the BiRealNet architecture for 1W1A, as described in [Xu et al., 2021b, Qin et al., 2020]. For 1W32A, we used the original ResNet18 architecture, following common practices in

Table 4: Performance comparison by testing accuracy of optimizers on ImageNet dataset across ResNet18 architectures. W/A denotes the bit-width of weights and activations.

| Methods | W/A | ResNet18 | |
|---|---|---|---|
| | | Top1 (%) | Top5 (%) |
| Bop Helwegen et al. [2019] | 1/1 | 54.2 | 77.2 |
| Bi-RealNet Liu et al. [2018] | 1/1 | 56.4 | 79.5 |
| IR-Net Qin et al. [2020] | 1/1 | 58.1 | 80.0 |
| BONN Gu et al. [2019b] | 1/1 | 59.3 | 81.6 |
| LCR-BNN Shang et al. [2022a] | 1/1 | 59.6 | 81.6 |
| SiBNN Wang et al. [2020] | 1/1 | 59.7 | 81.8 |
| SiMaN Lin et al. [2022] | 1/1 | 60.1 | 82.3 |
| md-tanh-s Ajanthan et al. [2021] | 1/1 | 60.3 | 82.3 |
| EqualBits Li et al. [2022] | 1/1 | 60.4 | 82.9 |
| DIR-Net Qin et al. [2023] | 1/1 | 60.4 | 81.9 |
| ReSTE Wu et al. [2023] | 1/1 | 60.9 | 82.6 |
| ReCU Xu et al. [2021b] | 1/1 | 61.0 | 82.6 |
| BiPer Vargas et al. [2024] | 1/1 | 61.4 | 83.1 |
| ReBNN Xu et al. [2023] | 1/1 | 61.6 | **83.4** |
| VISPA (Ours) | 1/1 | **62.1** | **83.4** |
| XNOR-Net Rastegari et al. [2016] | 1/32 | 60.8 | 83.0 |
| HWGQ Cai et al. [2017] | 1/32 | 61.3 | 83.2 |
| ADMM Leng et al. [2018] | 1/32 | 64.8 | 86.2 |
| IR-Net Qin et al. [2020] | 1/32 | 66.5 | 86.8 |
| Quantization networks Yang et al. [2019] | 1/32 | 66.5 | 87.3 |
| LCR-BNN Shang et al. [2022a] | 1/32 | 66.9 | 86.4 |
| ReSTE Wu et al. [2023] | 1/32 | 67.4 | 87.2 |
| DIR-Net Qin et al. [2023] | 1/32 | 67.5 | **87.9** |
| VISPA (Ours) | 1/32 | **68.2** | 87.8 |

Table 5: Performance comparison by testing accuracy (%) with and without $Z$ across CIFAR-10, CIFAR-100, and Tiny-ImageNet on VGG16 and ResNet18.

| Methods | CIFAR-10 | | CIFAR-100 | | Tiny-ImageNet |
|---|---|---|---|---|---|
| | VGG16 | ResNet18 | VGG16 | ResNet18 | ResNet18 |
| VISPA wo Z (Ours) | $92.85 \pm 0.09$ | $95.03 \pm 0.08$ | $70.12 \pm 0.10$ | $76.23 \pm 0.36$ | $56.73 \pm 0.33$ |
| VISPA (Ours) | $\mathbf{93.25 \pm 0.11}$ | $\mathbf{95.05 \pm 0.10}$ | $\mathbf{72.09 \pm 0.17}$ | $\mathbf{77.05 \pm 0.41}$ | $\mathbf{58.98 \pm 0.28}$ |

related literature [Qin et al., 2020, Shang et al., 2022a]. Additionally, for ResNet18, we kept the first, last, and down-sampling layers in full precision.

Table 3 and Table 4 present a performance comparison of various binarized neural network methods on the ImageNet dataset using AlexNet and ResNet18 architectures separately. For AlexNet, our method achieves state-of-the-art performance with a Top-1 accuracy of 51.1%, surpassing previous best results from Quantization Networks [Yang et al., 2019] by 3.2% in Top-1 accuracy. Similarly, in the 1W32A configuration, our method outperforms all others with a Top-1 accuracy of 59.4% and a Top-5 accuracy of 81.1%, demonstrating a significant improvement over the next best method, Quantization Networks [Yang et al., 2019].

For ResNet18, our method again sets new benchmarks with a Top-1 accuracy of 62.1%, which is 0.5% higher in Top-1 accuracy compared to the best previous method, ReBNN [Xu et al., 2023]. In the 1W32A configuration, our method achieves the highest Top-1 accuracy of 68.2% and a Top-5 accuracy of 87.8%, indicating its robustness and superior performance. These results highlight once again the effectiveness of our approach in improving the accuracy of BNNs across different architectures and configurations on complex datasets like ImageNet.

### 4.1 Ablation Studies

**Impact of Deviation Matrix $Z$** To investigate the significance of maintaining the correlation between weights, we compare VISPA with the simpler, correlation-free algorithm of Equation 3 with $\sigma = 0$ on CIFAR-10, CIFAR-100, and Tiny-ImageNet with the 1W32A setting. Table 5 provides the experimental results. It can be seen that VISPA consistently performs better across all configurations,

Figure 1: Impact of $K$ on model accuracy. The table shows the mean of testing accuracy and standard deviation for different $K$ across models and datasets. Darker colors indicate higher accuracy.

| Configurations | K=1 | K=2 | K=4 | K=6 | K=8 | K=10 |
|---|---|---|---|---|---|---|
| ResNet18 + CIFAR-10 (1W1A) | 92.53 ± 0.11 | 92.62 ± 0.13 | 92.77 ± 0.17 | 92.86 ± 0.07 | 92.83 ± 0.19 | 93.01 ± 0.07 |
| VGG-Small + CIFAR-10 (1W1A) | 92.72 ± 0.09 | 92.73 ± 0.13 | 92.67 ± 0.14 | 92.62 ± 0.12 | 92.55 ± 0.05 | 92.23 ± 0.09 |
| ResNet18 + CIFAR-10 (1W32A) | 95.01 ± 0.13 | 95.03 ± 0.18 | 95.02 ± 0.10 | 95.06 ± 0.12 | 95.05 ± 0.10 | 95.00 ± 0.14 |
| ResNet18 + CIFAR-100 (1W32A) | 76.45 ± 0.15 | 76.62 ± 0.12 | 77.04 ± 0.34 | 76.66 ± 0.24 | 77.05 ± 0.41 | 76.87 ± 0.40 |
| VGG16 + CIFAR-10 (1W32A) | 93.02 ± 0.20 | 93.21 ± 0.14 | 93.24 ± 0.09 | 93.34 ± 0.19 | 93.25 ± 0.11 | 93.30 ± 0.10 |
| VGG16 + CIFAR-100 (1W32A) | 71.09 ± 0.27 | 71.18 ± 0.11 | 71.83 ± 0.32 | 72.16 ± 0.10 | 72.09 ± 0.17 | 72.26 ± 0.18 |

especially on the more complex dataset Tiny-ImageNet, where the accuracy increases from 56.73 % to 58.89% for ResNet18. This suggests that the deviation matrix $Z$ might be particularly useful in scenarios with a higher number of classes and potentially more complex patterns.

**Impact of Covariance Rank $K$** We investigate the impact of the covariance rank $K$ on the accuracy of various datasets and models. Experiments are conducted on CIFAR-10 and CIFAR-100 datasets using configurations of 1W32A and 1W1A across architectures including VGG16, VGG-Small, and ResNet18, by setting different values of $K$ and performing 5 runs. Figure 1 presents the results. The data reveals that ResNet18 and VGG16 generally benefit from increasing $K$ values, with ResNet18 + CIFAR-10 (1W1A) and VGG16 + CIFAR-100 (1W32A) peaking at $K = 10$. The accuracy of ResNet18 on CIFAR-10 (1W1A) increases with higher $K$ values, from $92.53 \pm 0.11\%$ at $K = 1$ to $93.01 \pm 0.07\%$ at $K = 10$, indicating improved performance and stability. Conversely, VGG-Small on CIFAR-10 (1W1A) experiences a slight decline in accuracy with higher $K$. ResNet18 on CIFAR-100 (1W32A) shows the most variability, with accuracy peaking at $77.05 \pm 0.41\%$ for $K = 8$, suggesting an unpredictable impact of $K$. These results highlight that higher $K$ values generally improve accuracy and stability for most models, particularly for ResNet18 and VGG16, but the benefits may vary depending on the specific model and dataset configuration. We have two hypotheses for this lack of conclusive evidence: i) higher values of $K$ yield a larger feasible space and require a longer time to converge; ii) higher values of yield a more complex distribution from which it is more expensive to sample good weights at the inference stage.

## 5 Limitations and Open Problems

Our contributions naturally open a number of directions for further research. On the theoretical side, it would be interesting to efficiently implement the rounding suggested by Grothendieck's inequality. Similarly, we believe that the gradient descent approach of Burer-Monteiro can be formally analyzed as a gradient flow over the Bures-Wasserstein (BW) manifold of Gaussian distributions, which has recently been applied successfully in the context of variational inference [Lambert et al., 2022], to show that our method, with a proper choice of step size, converges to a stationary point of Problem 2 in the BW geometry.

On the experimental side, at the inference stage, our method currently draws and rounds 40 samples from the computed Gaussian distribution and averages the results. It is an active area of focus to further reduce the impact of this procedure on inference time. In preliminary results, we find that the process may be sped by using a smaller number of correlated samples via Gaussian quadrature. In this case, $2K + 1$ samples would suffice, which is as small as 3 for the $K = 1$ version of our method. Here we note that the prime runtime concern is to reduce the cost of training, which we achieve by binarizing both weights and activations while outperforming competitors.

Finally, we believe that the application of our method to the binarization of transformer architectures [He et al., 2023, Zhang et al., 2024] and to the general quantization settings, where the weights can take on more than two values, could have substantial practical consequences.

## 6 Acknowledgments

This research is supported by the Agency for Science, Technology, and Research (A*STAR) under its MTC Programmatic Fund M23L7b0021. And this work is partially supported by the National Natural Science Foundation of China under Grant (Nos. 62272093, 62137001).

# References

Thalaiyasingam Ajanthan, Puneet Dokania, Richard Hartley, and Philip Torr. Proximal Mean-Field for Neural Network Quantization. In *2019 IEEE/CVF International Conference on Computer Vision (ICCV)*, pages 4870–4879, October 2019.

Thalaiyasingam Ajanthan, Kartik Gupta, Philip Torr, Richad Hartley, and Puneet Dokania. Mirror descent view for neural network quantization. In *International conference on artificial intelligence and statistics*, pages 2809–2817. PMLR, 2021.

Milad Alizadeh, Javier Fernández-Marqués, Nicholas D. Lane, and Yarin Gal. An Empirical study of Binary Neural Networks' Optimisation. In *International Conference on Learning Representations*, September 2018.

Noga Alon and Assaf Naor. Approximating the Cut-Norm via Grothendieck's Inequality. *SIAM Journal on Computing*, 35(4):787–803, January 2006. ISSN 0097-5397. doi: 10.1137/S0097539704441629.

Alexander G. Anderson and Cory P. Berg. The High-Dimensional Geometry of Binary Neural Networks, May 2017.

Alexei Baevski, Yuhao Zhou, Abdelrahman Mohamed, and Michael Auli. wav2vec 2.0: A framework for self-supervised learning of speech representations. *Advances in neural information processing systems*, 33:12449–12460, 2020.

Yu Bai, Yu-Xiang Wang, and Edo Liberty. Proxquant: Quantized neural networks via proximal operators. *arXiv preprint arXiv:1810.00861*, 2018.

Boaz Barak and David Steurer. Proofs, beliefs and algorithms through the lens of Sum of Squares. https://www.sumofsquares.org/public/index.html, 2024.

Burak Bartan and Mert Pilanci. Training Quantized Neural Networks to Global Optimality via Semidefinite Programming. In *Proceedings of the 38th International Conference on Machine Learning*, pages 694–704. PMLR, July 2021.

Yoshua Bengio, Nicholas Léonard, and Aaron Courville. Estimating or propagating gradients through stochastic neurons for conditional computation. *arXiv preprint arXiv:1308.3432*, 2013.

Samuel Burer and Renato D.C. Monteiro. Local Minima and Convergence in Low-Rank Semidefinite Programming. *Mathematical Programming*, 103(3):427–444, July 2005. ISSN 1436-4646. doi: 10.1007/s10107-004-0564-1.

Zhaowei Cai, Xiaodong He, Jian Sun, and Nuno Vasconcelos. Deep learning with low precision by half-wave gaussian quantization. In *Proceedings of the IEEE conference on computer vision and pattern recognition*, pages 5918–5926, 2017.

Matthieu Courbariaux, Yoshua Bengio, and Jean-Pierre David. Binaryconnect: Training deep neural networks with binary weights during propagations. *Advances in neural information processing systems*, 28, 2015a.

Matthieu Courbariaux, Yoshua Bengio, and Jean-Pierre David. Training deep neural networks with low precision multiplications, September 2015b.

Matthieu Courbariaux, Itay Hubara, Daniel Soudry, Ran El-Yaniv, and Yoshua Bengio. Binarized neural networks: Training deep neural networks with weights and activations constrained to+ 1 or-1. *arXiv preprint arXiv:1602.02830*, 2016.

Jia Deng, Wei Dong, Richard Socher, Li-Jia Li, Kai Li, and Li Fei-Fei. Imagenet: A large-scale hierarchical image database. In *2009 IEEE conference on computer vision and pattern recognition*, pages 248–255. Ieee, 2009.

Jacob Devlin, Ming-Wei Chang, Kenton Lee, and Kristina Toutanova. Bert: Pre-training of deep bidirectional transformers for language understanding. *arXiv preprint arXiv:1810.04805*, 2018.

Ruizhou Ding, Ting-Wu Chin, Zeye Liu, and Diana Marculescu. Regularizing activation distribution for training binarized deep networks. In *Proceedings of the IEEE/CVF conference on computer vision and pattern recognition*, pages 11408–11417, 2019.

Sacha Friedli and Yvan Velenik. *Statistical Mechanics of Lattice Systems: A Concrete Mathematical Introduction*. Cambridge University Press, 1 edition, November 2017. ISBN 978-1-107-18482-4 978-1-316-88260-3. doi: 10.1017/9781316882603.

Dani Gamerman and Hedibert F. Lopes. *Markov Chain Monte Carlo*. Chapman and Hall/CRC, Boca Raton, 2nd edition edition, May 2006. ISBN 978-1-58488-587-0.

Ryan J Giordano, Tamara Broderick, and Michael I Jordan. Linear response methods for accurate covariance estimates from mean field variational bayes. *Advances in neural information processing systems*, 28, 2015.

Xavier Glorot and Yoshua Bengio. Understanding the difficulty of training deep feedforward neural networks. In *Proceedings of the thirteenth international conference on artificial intelligence and statistics*, pages 249–256. JMLR Workshop and Conference Proceedings, 2010.

Michel X Goemans and David P Williamson. Improved approximation algorithms for maximum cut and satisfiability problems using semidefinite programming. *Journal of the ACM (JACM)*, 42(6): 1115–1145, 1995.

A. Grothendieck. Résumé de la théorie métrique des produits tensoriels topologiques. *Bol. Soc. Mat. São Paulo*, 8:1–79, 1953.

Jiaxin Gu, Ce Li, Baochang Zhang, Jungong Han, Xianbin Cao, Jianzhuang Liu, and David Doermann. Projection convolutional neural networks for 1-bit cnns via discrete back propagation. In *Proceedings of the AAAI conference on artificial intelligence*, volume 33, pages 8344–8351, 2019a.

Jiaxin Gu, Junhe Zhao, Xiaolong Jiang, Baochang Zhang, Jianzhuang Liu, Guodong Guo, and Rongrong Ji. Bayesian optimized 1-bit cnns. In *Proceedings of the IEEE/CVF international conference on computer vision*, pages 4909–4917, 2019b.

Kai Han, Yunhe Wang, Yixing Xu, Chunjing Xu, Enhua Wu, and Chang Xu. Training binary neural networks through learning with noisy supervision. In *International conference on machine learning*, pages 4017–4026. PMLR, 2020.

Kaiming He, Xiangyu Zhang, Shaoqing Ren, and Jian Sun. Deep residual learning for image recognition. In *Proceedings of the IEEE conference on computer vision and pattern recognition*, pages 770–778, 2016.

Kaiming He, Georgia Gkioxari, Piotr Dollár, and Ross Girshick. Mask r-cnn. In *Proceedings of the IEEE international conference on computer vision*, pages 2961–2969, 2017.

Yefei He, Zhenyu Lou, Luoming Zhang, Jing Liu, Weijia Wu, Hong Zhou, and Bohan Zhuang. Bivit: Extremely compressed binary vision transformers. In *Proceedings of the IEEE/CVF International Conference on Computer Vision*, pages 5651–5663, 2023.

Koen Helwegen, James Widdicombe, Lukas Geiger, Zechun Liu, Kwang-Ting Cheng, and Roeland Nusselder. Latent weights do not exist: Rethinking binarized neural network optimization. *Advances in neural information processing systems*, 32, 2019.

Itay Hubara, Matthieu Courbariaux, Daniel Soudry, Ran El-Yaniv, and Yoshua Bengio. Binarized Neural Networks. In *Advances in Neural Information Processing Systems*, volume 29, 2016.

Minyoung Huh, Brian Cheung, Pulkit Agrawal, and Phillip Isola. Straightening out the straight-through estimator: Overcoming optimization challenges in vector quantized networks. In *International Conference on Machine Learning*, pages 14096–14113. PMLR, 2023.

Minje Kim and Paris Smaragdis. Bitwise neural networks. *arXiv preprint arXiv:1601.06071*, 2016.

Alex Krizhevsky, Geoffrey Hinton, et al. Learning multiple layers of features from tiny images. 2009.

Alex Krizhevsky, Ilya Sutskever, and Geoffrey E Hinton. Imagenet classification with deep convolutional neural networks. *Advances in neural information processing systems*, 25, 2012.

Marc Lambert, Sinho Chewi, Francis Bach, Silvère Bonnabel, and Philippe Rigollet. Variational inference via wasserstein gradient flows. *Advances in Neural Information Processing Systems*, 2022.

Huu Le, Rasmus Kjær Høier, Che-Tsung Lin, and Christopher Zach. Adaste: An adaptive straight-through estimator to train binary neural networks. In *Proceedings of the IEEE/CVF Conference on Computer Vision and Pattern Recognition*, pages 460–469, 2022.

Ya Le and Xuan Yang. Tiny imagenet visual recognition challenge. *CS 231N*, 7(7):3, 2015.

Cong Leng, Zesheng Dou, Hao Li, Shenghuo Zhu, and Rong Jin. Extremely low bit neural network: Squeeze the last bit out with admm. In *Proceedings of the AAAI Conference on Artificial Intelligence*, volume 32, 2018.

Yunqiang Li, Silvia-Laura Pintea, and Jan C van Gemert. Equal bits: Enforcing equally distributed binary network weights. In *Proceedings of the AAAI conference on artificial intelligence*, volume 36, pages 1491–1499, 2022.

Mingbao Lin, Rongrong Ji, Zihan Xu, Baochang Zhang, Yan Wang, Yongjian Wu, Feiyue Huang, and Chia-Wen Lin. Rotated binary neural network. *Advances in neural information processing systems*, 33:7474–7485, 2020.

Mingbao Lin, Rongrong Ji, Zihan Xu, Baochang Zhang, Fei Chao, Chia-Wen Lin, and Ling Shao. Siman: Sign-to-magnitude network binarization. *IEEE Transactions on Pattern Analysis and Machine Intelligence*, 45(5):6277–6288, 2022.

Liyuan Liu, Chengyu Dong, Xiaodong Liu, Bin Yu, and Jianfeng Gao. Bridging discrete and backpropagation: Straight-through and beyond. *Advances in Neural Information Processing Systems*, 36, 2024.

Zechun Liu, Baoyuan Wu, Wenhan Luo, Xin Yang, Wei Liu, and Kwang-Ting Cheng. Bi-real net: Enhancing the performance of 1-bit cnns with improved representational capability and advanced training algorithm. In *Proceedings of the European conference on computer vision (ECCV)*, pages 722–737, 2018.

Zechun Liu, Zhiqiang Shen, Marios Savvides, and Kwang-Ting Cheng. Reactnet: Towards precise binary neural network with generalized activation functions. In *Computer Vision–ECCV 2020: 16th European Conference, Glasgow, UK, August 23–28, 2020, Proceedings, Part XIV 16*, pages 143–159. Springer, 2020.

Brais Martinez, Jing Yang, Adrian Bulat, and Georgios Tzimiropoulos. Training binary neural networks with real-to-binary convolutions. *arXiv preprint arXiv:2003.11535*, 2020.

Xiangming Meng, Roman Bachmann, and Mohammad Emtiyaz Khan. Training Binary Neural Networks using the Bayesian Learning Rule. In *Proceedings of the 37th International Conference on Machine Learning*, pages 6852–6861. PMLR, November 2020.

Paul Merolla, Rathinakumar Appuswamy, John Arthur, Steve K. Esser, and Dharmendra Modha. Deep neural networks are robust to weight binarization and other non-linear distortions, June 2016.

Lorenz K. Muller and Giacomo Indiveri. Rounding Methods for Neural Networks with Low Resolution Synaptic Weights, April 2015.

Adam Paszke, Sam Gross, Francisco Massa, Adam Lerer, James Bradbury, Gregory Chanan, Trevor Killeen, Zeming Lin, Natalia Gimelshein, Luca Antiga, et al. Pytorch: An imperative style, high-performance deep learning library. *Advances in neural information processing systems*, 32, 2019.

Haotong Qin, Ruihao Gong, Xianglong Liu, Mingzhu Shen, Ziran Wei, Fengwei Yu, and Jingkuan Song. Forward and backward information retention for accurate binary neural networks. In *Proceedings of the IEEE/CVF conference on computer vision and pattern recognition*, pages 2250–2259, 2020.

Haotong Qin, Xiangguo Zhang, Ruihao Gong, Yifu Ding, Yi Xu, and Xianglong Liu. Distribution-sensitive information retention for accurate binary neural network. *International Journal of Computer Vision*, 131(1):26–47, 2023.

Mohammad Rastegari, Vicente Ordonez, Joseph Redmon, and Ali Farhadi. XNOR-Net: ImageNet Classification Using Binary Convolutional Neural Networks, August 2016.

Hiroki Sayama. *Introduction to the Modeling and Analysis of Complex Systems*. Open SUNY Textbooks. Published by Open SUNY Textbooks, Milne Library, State University of New York at Geneseo, Geneseo, NY, 2015. ISBN 978-1-942341-06-2.

Gu Shan, Zhang Guoyin, Jia Chengwei, and Wu Yanxia. Sgdat: An optimization method for binary neural networks. *Neurocomputing*, page 126431, 2023.

Yuzhang Shang, Dan Xu, Bin Duan, Ziliang Zong, Liqiang Nie, and Yan Yan. Lipschitz continuity retained binary neural network. In *European conference on computer vision*, pages 603–619. Springer, 2022a.

Yuzhang Shang, Dan Xu, Ziliang Zong, Liqiang Nie, and Yan Yan. Network binarization via contrastive learning. In *European Conference on Computer Vision*, pages 586–602. Springer, 2022b.

Mingzhu Shen, Xianglong Liu, Ruihao Gong, and Kai Han. Balanced binary neural networks with gated residual. In *ICASSP 2020-2020 IEEE International Conference on Acoustics, Speech and Signal Processing (ICASSP)*, pages 4197–4201. IEEE, 2020.

Karen Simonyan and Andrew Zisserman. Very deep convolutional networks for large-scale image recognition. *arXiv preprint arXiv:1409.1556*, 2014.

Cuauhtemoc Daniel Suarez-Ramirez, Miguel Gonzalez-Mendoza, Leonardo Chang, Gilberto Ochoa-Ruiz, and Mario Alberto Duran-Vega. A bop and beyond: a second order optimizer for binarized neural networks. In *Proceedings of the IEEE/CVF Conference on Computer Vision and Pattern Recognition*, pages 1273–1281, 2021.

Ilya Sutskever, James Martens, George Dahl, and Geoffrey Hinton. On the importance of initialization and momentum in deep learning. In *International conference on machine learning*, pages 1139–1147. PMLR, 2013.

Vivienne Sze, Yu-Hsin Chen, Tien-Ju Yang, and Joel S Emer. Efficient processing of deep neural networks: A tutorial and survey. *Proceedings of the IEEE*, 105(12):2295–2329, 2017.

Wei Tang, Gang Hua, and Liang Wang. How to train a compact binary neural network with high accuracy? In *Proceedings of the AAAI conference on artificial intelligence*, volume 31, 2017.

Neil C. Thompson, Kristjan Greenewald, Keeheon Lee, and Gabriel F. Manso. The Computational Limits of Deep Learning, July 2022.

Yaman Umuroglu, Nicholas J Fraser, Giulio Gambardella, Michaela Blott, Philip Leong, Magnus Jahre, and Kees Vissers. Finn: A framework for fast, scalable binarized neural network inference. In *Proceedings of the 2017 ACM/SIGDA international symposium on field-programmable gate arrays*, pages 65–74, 2017.

Lieven Vandenberghe and Stephen Boyd. Semidefinite programming. *SIAM Review*, 38(1):49–95, 1996. doi: 10.1137/1038003.

Antoine Vanderschueren and Christophe De Vleeschouwer. Are straight-through gradients and soft-thresholding all you need for sparse training? In *Proceedings of the IEEE/CVF Winter Conference on Applications of Computer Vision*, pages 3808–3817, 2023.

Edwin Vargas, Claudia Correa, Carlos Hinojosa, and Henry Arguello. Biper: Binary neural networks using a periodic function. *arXiv preprint arXiv:2404.01278*, 2024.

Vijay V. V. Vazirani. *Approximation Algorithms*. Springer, Berlin Heidelberg, December 2010. ISBN 978-3-642-08469-0.

Martin J Wainwright, Michael I Jordan, et al. Graphical models, exponential families, and variational inference. *Foundations and Trends® in Machine Learning*, 1(1–2):1–305, 2008.

Peisong Wang, Xiangyu He, Gang Li, Tianli Zhao, and Jian Cheng. Sparsity-inducing binarized neural networks. In *Proceedings of the AAAI conference on artificial intelligence*, volume 34, pages 12192–12199, 2020.

Ronald J. Williams. Simple statistical gradient-following algorithms for connectionist reinforcement learning. *Machine Learning*, 8(3):229–256, May 1992. ISSN 1573-0565. doi: 10.1007/BF00992696.

Xiao-Ming Wu, Dian Zheng, Zuhao Liu, and Wei-Shi Zheng. Estimator meets equilibrium perspective: A rectified straight through estimator for binary neural networks training. In *Proceedings of the IEEE/CVF International Conference on Computer Vision*, pages 17055–17064, 2023.

Sheng Xu, Yanjing Li, Teli Ma, Mingbao Lin, Hao Dong, Baochang Zhang, Peng Gao, and Jinhu Lu. Resilient binary neural network. In *Proceedings of the AAAI Conference on Artificial Intelligence*, volume 37, pages 10620–10628, 2023.

Yixing Xu, Kai Han, Chang Xu, Yehui Tang, Chunjing Xu, and Yunhe Wang. Learning frequency domain approximation for binary neural networks. *Advances in Neural Information Processing Systems*, 34:25553–25565, 2021a.

Zihan Xu, Mingbao Lin, Jianzhuang Liu, Jie Chen, Ling Shao, Yue Gao, Yonghong Tian, and Rongrong Ji. Recu: Reviving the dead weights in binary neural networks. In *Proceedings of the IEEE/CVF international conference on computer vision*, pages 5198–5208, 2021b.

Ping Xue, Yang Lu, Jingfei Chang, Xing Wei, and Zhen Wei. Self-distribution binary neural networks. *Applied Intelligence*, 52(12):13870–13882, 2022.

Jiwei Yang, Xu Shen, Jun Xing, Xinmei Tian, Houqiang Li, Bing Deng, Jianqiang Huang, and Xian-sheng Hua. Quantization networks. In *Proceedings of the IEEE/CVF conference on computer vision and pattern recognition*, pages 7308–7316, 2019.

Penghang Yin, Jiancheng Lyu, Shuai Zhang, Stanley Osher, Yingyong Qi, and Jack Xin. Understanding straight-through estimator in training activation quantized neural nets. *arXiv preprint arXiv:1903.05662*, 2019.

Alp Yurtsever, Joel A. Tropp, Olivier Fercoq, Madeleine Udell, and Volkan Cevher. Scalable Semidefinite Programming. *SIAM Journal on Mathematics of Data Science*, 3(1):171–200, January 2021. doi: 10.1137/19M1305045.

Dongqing Zhang, Jiaolong Yang, Dongqiangzi Ye, and Gang Hua. Lq-nets: Learned quantization for highly accurate and compact deep neural networks. In *Proceedings of the European conference on computer vision (ECCV)*, pages 365–382, 2018.

Yichi Zhang, Ankush Garg, Yuan Cao, Lukasz Lew, Behrooz Ghorbani, Zhiru Zhang, and Orhan Firat. Binarized neural machine translation. *Advances in Neural Information Processing Systems*, 36, 2024.

Chenzhuo Zhu, Song Han, Huizi Mao, and William J Dally. Trained ternary quantization. *arXiv preprint arXiv:1612.01064*, 2016.

# A  Appendix

## A.1  Proofs

*Proof of Theorem 1.*  Both proofs are based on the change of variable:

$$\boldsymbol{w} = \boldsymbol{\mu} + \boldsymbol{Z}\boldsymbol{r}$$

where $\boldsymbol{r} \sim \mathcal{N}(0, \boldsymbol{I}_K)$. For the gradient with respect to the mean vector $\mu$, we have:

$$\nabla_{\boldsymbol{\mu}}\mathbb{E}_{\boldsymbol{w},\boldsymbol{x}}[L(f(\boldsymbol{x},\boldsymbol{w}),y_{\boldsymbol{x}})] = \mathbb{E}_{\boldsymbol{x}}\left[ \int_{\mathbb{R}^K} (2\pi)^{-K/2}\, \nabla_{\boldsymbol{\mu}}L(f(\boldsymbol{x},\boldsymbol{\mu}+\boldsymbol{Z}\boldsymbol{r}),y_{\boldsymbol{x}})\, e^{-\|\boldsymbol{r}\|^2/2}dr \right]$$

$$= \mathbb{E}_{\boldsymbol{w},\boldsymbol{x}}[\nabla_{\boldsymbol{w}}L(f(\boldsymbol{x},\boldsymbol{w}),y_{\boldsymbol{x}})]$$

For the gradient with respect to the deviation matrix $\boldsymbol{Z}$, we have:

$$\nabla_{\boldsymbol{Z}}\mathbb{E}_{\boldsymbol{w},\boldsymbol{x}}[L(f(\boldsymbol{x},\boldsymbol{w}),y_{\boldsymbol{x}})] = \mathbb{E}_{\boldsymbol{x}}\left[ \int_{\mathbb{R}^K} (2\pi)^{-K/2}\, \nabla_{\boldsymbol{Z}}L(f(\boldsymbol{x},\boldsymbol{\mu}+\boldsymbol{Z}\boldsymbol{r}),y_{\boldsymbol{x}})\, e^{-\|\boldsymbol{r}\|^2/2}dr \right]$$

$$= \mathbb{E}_{\boldsymbol{x}}\left[ \int_{\mathbb{R}^K} (2\pi)^{-K/2}\, \nabla_{\boldsymbol{w}}[L(f(\boldsymbol{x},\boldsymbol{w}),y_{\boldsymbol{x}})\,\boldsymbol{r}^T]|_{\boldsymbol{w}=\boldsymbol{Z}+\boldsymbol{r}}\, e^{-\|\boldsymbol{r}\|^2/2}dr \right]$$

$$= \mathbb{E}_{\boldsymbol{r},\boldsymbol{x}}[\nabla_{\boldsymbol{w}}[L(f(\boldsymbol{x},\boldsymbol{w}),y_{\boldsymbol{x}})\boldsymbol{r}^T]|_{\boldsymbol{w}=\boldsymbol{Z}\boldsymbol{r}+\boldsymbol{\mu}}],$$

$\square$

## A.2  Impact of the Initialization of $Z$ on the Performance

We draw $\boldsymbol{Z}$ from the Xavier Normal Initialization [Glorot and Bengio, 2010] using a normal distribution with a mean of 0 and a standard deviation given by

$$\sigma = s * \sqrt{\frac{2}{\text{fan\_in} + \text{fan\_out}}}. \tag{4}$$

where fan_in is the number of input units, fan_out is the number of output units and $s$ is the scaling factor. To investigate the impact of $s$ to the accuracy, we conduct experiments on CIFAR-10 and CIFAR-100 dataset with the commonly deployed architectures VGG-17 and ResNet18 for the 1W32A setting by setting the covariance rank $K$ at 8 with various $s$ values, specifically at $s = 1, 5, 10, 15, 20, 25$ by five runs. Figure 2 shows the result. We observe that the accuracy of the models typically peaks at $s = 5$ or $s = 10$. Besides, the standard deviation is relatively low across all $s$ values, indicating consistent performance. Finally, we set $s = 10$ across all experiments in Section 4.

Figure 2: Impact of the initailization of $\boldsymbol{Z}$ on model accuracy in terms of the hyper-parameter $s$. The table shows the mean accuracy and standard deviation for different values of $s$ across models and datasets. Darker colors indicate higher accuracy values. To help reviewers easily see differences and peak values, we prefer visualizing in table rather than in figure. On small datasets like CIFAR, visualizing differences in figures is difficult due to minimal differences. For example, the experimental results of CIFAR-10 + VGG-16 (1W32A) only has 0.1% difference between $s = 5$ and $s = 25$.

| Configurations | s=1 | s=5 | s=10 | s=15 | s=20 | s=25 |
|---|---|---|---|---|---|---|
| VGG-16 + CIFAR-100 (1W32A) | 1.48 ± 0.08 | 72.27 ± 0.17 | 72.09 ± 0.17 | 71.96 ± 0.14 | 71.62 ± 0.29 | 71.28 ± 0.22 |
| ResNet18 + CIFAR-100 (1W32A) | 70.93 ± 0.14 | 76.64 ± 0.15 | 77.05 ± 0.41 | 76.92 ± 0.13 | 76.40 ± 0.42 | 76.63 ± 0.25 |
| VGG-16 + CIFAR-10 (1W32A) | 87.94 ± 0.21 | 93.23 ± 0.09 | 93.25 ± 0.11 | 93.23 ± 0.02 | 93.22 ± 0.08 | 93.13 ± 0.08 |
| ResNet18 + CIFAR-10 (1W32A) | 91.95 ± 0.03 | 94.78 ± 0.08 | 95.05 ± 0.10 | 95.06 ± 0.07 | 95.06 ± 0.09 | 95.04 ± 0.15 |

